# Non-linear PI Control Inspired by Biological Control Systems

**Lyndon J. Brown**    **Gregory E. Gonye**    **James S. Schwaber** *
Experimental Station, E.I. DuPont deNemours & Co. Wilmington, DE 19880

## Abstract

A non-linear modification to PI control is motivated by a model of a signal transduction pathway active in mammalian blood pressure regulation. This control algorithm, labeled PII (proportional with intermittent integral), is appropriate for plants requiring exact set-point matching and disturbance attenuation in the presence of infrequent step changes in load disturbances or set-point. The proportional aspect of the controller is independently designed to be a disturbance attenuator and set-point matching is achieved by intermittently invoking an integral controller. The mechanisms observed in the Angiotensin II/AT1 signaling pathway are used to control the switching of the integral control. Improved performance over PI control is shown on a model of cyclopentenol production. A sign change in plant gain at the desirable operating point causes traditional PI control to result in an unstable system. Application of this new approach to this problem results in stable exact set-point matching for achievable set-points.

Biological processes have evolved sophisticated mechanisms for solving difficult control problems. By analyzing and understanding these natural systems it is possible that principles can be derived which are applicable to general control systems. This approach has already been the basis for the field of artificial neural networks, which are loosely based on a model of the electrical signaling of neurons. A suitable candidate system for analysis is blood pressure control. Tight control of blood pressure is critical for survival of an animal. Chronically high levels can lead to premature death. Low blood pressure can lead to oxygen and nutrient deprivation and sudden load changes must be quickly responded to or loss of consciousness can result. The baroreflex, reflexive change of heart rate in response to blood pressure challenge, has been previously studied in order to develop some insights into biological control systems [1, 2, 3].

Gregory.E.Gonye-PHD@usa.dupont.com James.S.Scwhaber@usa.dupont.com

Neurons exhibit complex dynamic behavior that is not directly revealed by their electrical behavior, but is incorporated in biochemical signal transduction pathways. This is an important basis for plasticity of neural networks. The area of the brain to which the baroreceptor afferents project is the nucleus of tractus solitarus (NTS). The neurons in the NTS are rich with diverse receptors for signaling pathways. It is logical that this richness and diversity play a crucial role in the signal processing that occurs here. Hormonal and neurotransmitter signals can activate signal transduction pathways in the cell, which result in physical modification of some components of a cell, or altered gene regulation. Fuxe et al [4] have shown the presence of the angiotensin II/AT1 receptor pathway in NTS neurons, and Herbert [5] has demonstrated its ability to affect the baroreflex.

To develop understanding of the effects of biochemical pathways, a detailed kinetic model of the angiotensin/AT1 pathway was developed. Certain features of this model and the baroreflex have interesting characteristics from a control engineering perspective. These features have been used to develop a novel control strategy. The resulting control algorithm utilizes a proportional controller that intermittently invokes integral action to achieve set-point matching. Thus the controller will be labeled PII.

The use of integral control is popular as it guarantees cancellation of offsets and ensures exact set-point matching. However, the use of integral control does have drawbacks. It introduces significant lag in the feedback system, which limits the bandwidth of the system. Increasing the integral gain, in order to improve response time, can lead to systems with excessive overshoot, excessive settling times, and less robustness to plant changes or uncertainty. Many processes in the chemical industry have a steady-state response curve with a maximum and frequently, the optimal operating condition is at this peak. Unfortunately, any controller with true integral action will be unstable at this operating point.

In a crude sense, the integrator learns the constant control action required to achieve set-point matching. If the integral control is viewed as a simple learning device, than a logical step is to remove it from the feedback loop once the necessary offset has been learned. If the offset is being successfully compensated for, only noise remains as a source for learning. It has been well established that learning based on nothing but noise leads to undesirable results. The maxim, 'garbage in, garbage out' will apply. Without integral control, the proportional controller can be made more aggressive while maintaining stability margins and/or control actions at similar levels. This control strategy will be appropriate for plants with infrequent step changes in set-points or loads. The challenge becomes deciding when, and how to perform this switching so that the resulting controller provides significant improvements.

## 1    Angiotensin II/AT1 receptor Signal Transduction Model

Regulation of blood pressure is a vital control problem in mammals. Blood pressure is sensed by stretch sensitive cells in the aortic arch and carotid sinus. These cells transmit signals to neurons in the NTS which are combined with other signals from the central nervous system (CNS) resulting in changes to the cardiac output and vascular tone [6]. This control is implemented by two parallel systems in the CNS, the sympathetic and parasympathetic nervous systems. The sympathetic system primarily affects the vascular tone and the parasympathetic system affects cardiac output [7]. Cardiac control can have a larger and faster effect, but long term application of this control is injurious to the overall health of the animal. Pottman et al [2] have suggested that these two systems separately control for long term set-point control and fast disturbance rejection.

One receptor in NTS neuronal cells is the AT1 receptor which binds Angiotensin II. The NTS is located in the brain stem where much of the processing of the autonomic regulatory systems reside. Angiotensin infusion in this region of the brain has been shown to significantly affect blood pressure control. In order to understand this aspect of neuronal behavior, a detailed kinetic model of this signaling pathway was developed. The pathway is presented in Figure 2. The outputs can be considered to be the concentrations of $G_q$·GTP, $G_{\beta\gamma}$, activated protein kinase C, and/or calmodulin dependent protein kinase.

Several reactions in the cascade are of interest. The binding of phospholipase C is significantly slower than the other steps in the reaction. This can be modeled as a first order transfer function with a long time constant or as a pure integrator. The $IP_3$ receptor is a ligand gated channel on the membrane of the endoplasmic reticulum (ER). As Figure 2 shows, when $IP_3$ binds to this receptor, calcium is released from the ER into the cells cytoplasm. However the $IP_3$ receptor also has 2 binding sites on its cytoplasmic domain for binding calcium. The first has relatively fast dynamics and causes a substantial increase in the channel opening. The second calcium binding site has slower dynamics and inactivates the channel. The effect of this first binding site is to introduce positive feedback into the model. In traditional control literature, positive feedback is generally undesirable. Thus it is very interesting to see positive feedback in neuronal control systems.

A typical surface response for the model, comparing the time response of activated calmodulin versus the peak concentration of a pulse of angiotensin, is shown in Figure 1. The results are consistent with behavior of cells measured by Li and Guyenet [8]. The output level is seen to abruptly rise after a delay, which is a decreasing function of the magnitude of the input. Unlike a linear system, both the magnitude and speed of the response of the system are functions of the magnitude of the input. Further, the relaxing of the system to its equilibrium is a very slow response as compared to its activation. This behavior can be attributed to the positive feedback response inherent to the IP3 receptor. The effect of the slow dynamics of the phospholipase C binding, and the IP3 receptor dynamics results in an activation behavior similar to a threshold detector on the integrated input signal. However, removal of the input results in a slow recovery back to zero. The activation of the calcium calmodulin dependent protein kinase can lead to phosphorilation of channels that result in synaptic conductance changes that are functionally related to the amount of activated kinase. The activation of calcium calmodulin can also lead to changes in gene regulation that could potentially result in long term changes in the neurons synaptic conductances.

## 2 Proportional with Intermittent Integral Control

Key features from the model that are incorporated in the control law are:

1. separate controllers for set-point control and disturbance attenuation;
2. activation of set-point controller when integrated error exceeds threshold;
3. strength of integral action when activated will be a function of the speed with which activation was achieved;
4. smooth removal of integral action, without disruption of control action.

The PII controller begins initially as a proportional controller with a nominal offset added to its output. The integrated error is monitored. The integral controller is turned on when the integrated error exceeds a threshold. Once the integral control action is activated, it remains active as long as the error is excessive. Once the error is not significant, then the integral control action can be removed in a

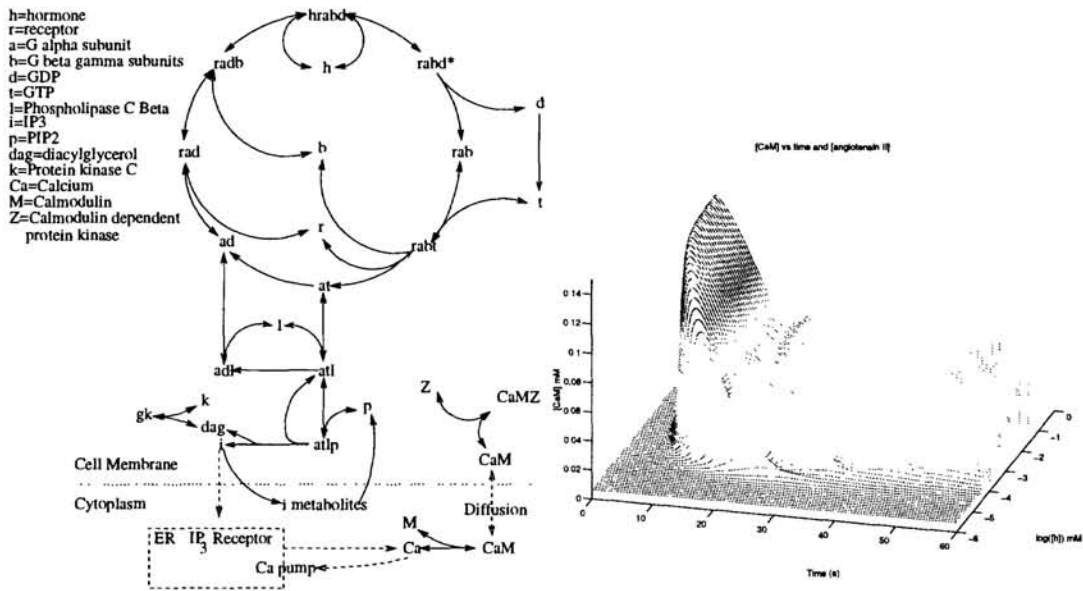

Figure 1: Schematic and Surface Responses of Angiotensin II / AT1 Model

smooth manner. This has been achieved by allowing the value of the integral gain, $K_i$, to decay exponentially. It is important that this is done in such a manner as not to affect the actual control signal. This can be achieved by adjusting the offset appropriately. Since $u = K_p e + K_i e/s$ and $\dot{K}_i \propto -K_i$, then $u$ can be made constant for constant $e$ by adding offset $K_o$ where $\dot{K}_o \propto K_i e/s$. The integral action is completely removed once $K_i$ has decayed to the point where it is no longer significant. In order to make the effect of activation of the integrator correspond to the behavior of the angiotensin model, the integrated error is scaled by the time spent reaching the threshold when the integrator is turned on. This corresponds to point 3 above.

If the error undergoes significant change when the integrator is already fully active the system will behave similarly to a system with a PI controller whose gains have been set too high. This may result in significant overshoot and possibly instability. There is a small chance that even with infrequent step changes, the residual error, or random disturbance could trigger the integrator immediately before a step change. In a biological control system, control does not rest in one neuron or necessarily in one signal transduction pathway but in multiple pathways. Furthermore, study of individual cells shows a great deal of variability in the details of their behavior. By implementing the intermittent integral control as a sum of many equivalent controllers, as in left side of Figure 2, with variability in their threshold parameters, a controller can be developed that is not subject to the chance of being fully activated by random disturbance or residual error. During steady-state operation these integrators will quickly deactivate when noise or small disturbances trigger them, as the error will be less than the threshold. However, an actual step change in the error signal will result in all or most of the integrators activating, and remaining active until the error is compensated for.

The block diagram on right side of Figure 2 and the time dependent definitions in Table 1 precisely define the control algorithm for the single integrator case.

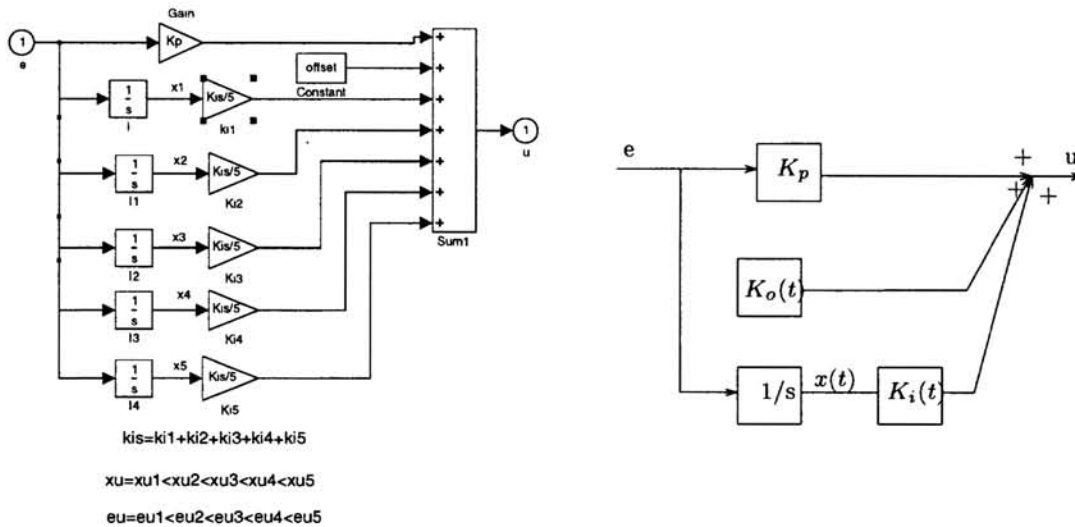

Figure 2: Block Diagrams for Control Algorithm Implementations

| If | then |
|---|---|
| $t = t_0$ | $x(t_0) = 0,\; K_i(t_0) = 0,\; K_o(t_0) = K_o^*,\; t_l(t_0) = t_0.$ |
| $K_i(t) = 0$ and $|x(t)| > x_u$ | $K_i(t) = K_i^*, x(t^+) = \dfrac{x(t)}{\max(1, K_s(t-t_l))}.$ |
| $|K_i(t)| > K_i^l$ and $|e(t)| < e_u$ | $\dot{K}_i(t) = -K_{decay}K_i(t),\; \dot{K}_o(t) = K_{decay}K_i(t)x(t).$ |
| $0 < |K_i(t)| < K_i^l$ | $K_o(t^+) = K_o(t) + K_i(t)x(t),$<br>$K_i(t^+) = 0,\; x(t^+) = 0,\; t_l(t^+) = t.$ |
| Otherwise | $\dot{K}_i(t) = 0,\quad \dot{K}_o(t) = 0,\quad \dot{x} = e,\quad \dot{t}_l(t) = 0.$ |

Table 1: Definition of Gains for PII Control

# 3 Control of CSTR Reactor for Cyclopentenol Production

The model of the CSTR reactor is taken from [9]. The basic process converts cyclopentadiene to cyclopentenol. Cyclopentenol can undergo a further undesirable reaction to form cyclopentadiol, and cyclopentadiene can undergo an alternative reaction to form dicyclopentadiene. The rates of the reactions are temperature dependent. Inputs to the model are flow rate, and the jacket temperature. The first input is the control input, and the jacket temperature is an unmeasured disturbance, with a root mean square deviation of 0.1 C about a nominal value of 130 C. The regulated output will be the cyclopentenol concentration in the outflow.

The steady-state response of this process is shown in Figure 3. Operation in the region labeled II up to the peak of the curve labeled VIII has been considered. At the point labeled VIII, the steady-state gain of the plant goes to 0. Plants with steady-state gains which change sign can not be stably controlled with PI control. An additional complicating factor is that the plant has significant inverse response in this region.

Criteria for this control design problem, in order of importance are

- operate between 45 and 60 l/hour with reasonable high frequency gain
- minimize the overshoot
- minimize rise time

- minimize the inverse response

Satisfying the first and last criteria should ensure a robust controller. Precise numerical performance criteria for the rise time have not been specified as no fixed values are reasonable for the entire region.

A PI controller, as well as a PII controller have been designed and the results are displayed in Figures 3. The controller parameters were $K_p = 75$, $K_i = 7500$ for the standard PI controller. The PII controller used 5 equally weighted parallel integrators with $K_p = 125$, total $K_i^* = 10000$ and $K_{decay} = 100$. The threshold parameters were chosen as $e_u = [4\ 3\ 2\ 1\ 1] * 0.00025$, $x_u = [16\ 8\ 4\ 2\ 1] * 0.00004$, and $K_i^l = K_i^*/5$

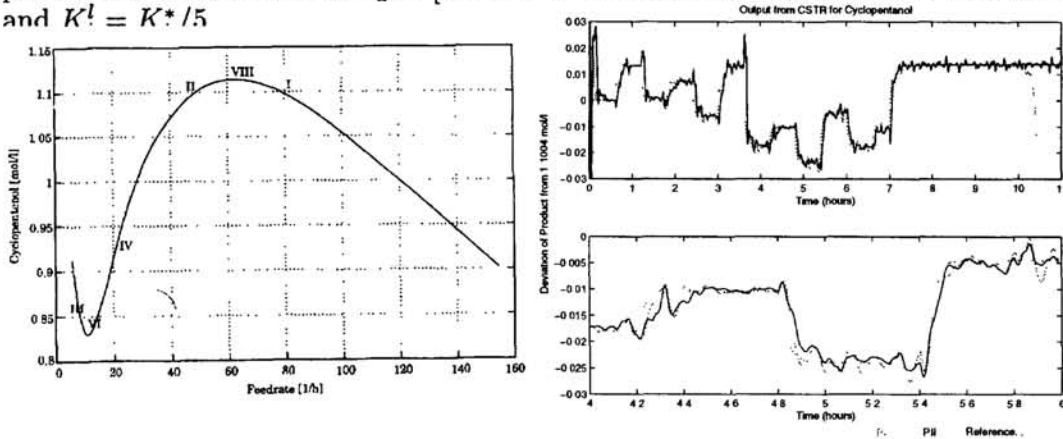

Figure 3: Steady-State Response of Cyclopentenol CSTR Reactor and Output Concentration from CSTR Reactor

The set-point was chosen to be a series of smoothed steps. Smoothing was performed with a first-order, low-pass filter with unity DC gain and a time constant of $30\ \text{hours}^{-1}$. While operating in the region of design from 0 to 4.8 hours and 5.4 to 7 hours, the PII controlled system, as compared to the PI controlled system, had reduced inverse response, less worst-case overshoot, similar response times and greater disturbance attenuation. A closer examination of the PI controlled system, during the interval 4.8-5.4 hours, showed that at this extreme operating point, oscillations of a fixed period begin to appear. This indicates the existence of poorly damped poles. The PII controlled system did not show this degradation of performance.

The set-point was raised to nearly the maximum achievable concentration. This allows examination of the behavior of the controller when operating near regions of uncertainty in the sign of the plant gain. This operating point achieves the maximum possible conversion to cyclopentenol and thus has significant economic advantages. In the region from 7.2s to 10s, there is a 10% reduction in the disturbance response with the PII controller. At this operating point, the PI controlled system can be shown to be locally stable. However, the effects of integrated noise easily allow the system trajectory to escape the region of attraction. As expected, the PI controlled system went unstable. The PII controlled system remains well behaved. The simulation was run for a total simulated time of 43 hours at this operating point, and repeated many times without seeing any loss of stability with PII controller. With PI control, the system went unstable within 10 hours for each trial. Thus, PII control allows operation at set-points closer to maximums or minimums.

## 4   Conclusion

The mechanisms that biological control systems employ to successfully control nonlinear, time varying, multivariable physiological systems under very demanding per-

formance requirements are likely to have application in process control problems. In addition to neural networks already incorporated in advanced controllers, cells process information through biochemical signal transduction networks that may also contain useful non-linear mechanisms. A model of one such pathway has been developed, and features have been identified which can be used to develop an improved control system.

The fundamental idea is to design two separate control laws, one intermittently used for cancelling infrequently changing but mostly predictable disturbances, and another for attenuating white disturbances. The first controller learns the simple characteristics of the predictable disturbance. When the predictable disturbance is learned, it can be canceled with an open loop controller, and no further learning takes place. However if it appears that the open loop controller is not cancelling the disturbance, further learning takes place until the disturbance is again successfully cancelled. The second controller is designed strictly for fast disturbance attenuation. Without the lag inherent in integration, the controller can be made more aggressive resulting in better performance. The two controllers can be integrated by applying the threshold and switching mechanisms identified in the signal transduction model.

## Footnotes

*lyndon.j.brown@usa.dupont.com          Address correspondence to this author

# References

[1] M. A. Henson, B. A. Ogunnaike, J. S. Schwaber, and F. J. Doyle III, "The baroreceptor reflex: A biological control system with applications in chemical process control," *I&EC Research*, vol. 33, pp. 2453–2465, 1994.

[2] M. Pottman, M. A. Henson, B. A. Ogunnaike, and J. S. Schwaber, "A parallel control strategy abstracted from the baroreceptor reflex," *Chemical Engineering Science*, vol. 51, pp. 931–945, 1996.

[3] H. S. Kwatra, F. J. Doyle III, and J. S. Schwaber, "Dynamic gain scheduled process control," *Chemical Engineering Science*, 1997.

[4] K. Fuxe and B. B. et al, "Pre- and post-synaptic features of the central angiotensin systems: Indications for a role of angiotensin peptides in volume transmission and for interactions with central monamine neurons," *Clin Exp Hypertens [Theory Pract]*, vol. A10, pp. 143–168, 1988.

[5] J. Herbert, "Studying the central actions of angiotensin using the expression of immediate-early genes: Expectations and limitations," *Regulatory Peptides*, vol. 66, pp. 13–18, 1996.

[6] K. M. Spyer, "The central nervous organization of reflex circulatory control," in *Clin Exp Hypertens [Theory Pract]Central Regulation of Automanomic Fuctions* (A. D. Loewy and K. M. Spyer, eds.), p. 168, New York: Oxford University Press, 1990.

[7] M. N. Kumada, N. Terui, and T. Kuwaki, "Arterial baroreceptor reflex: Its central and peripheral neural mechanisms," *Progr. Neurophysiol.*, vol. 35, p. 331, 1988.

[8] Y. Li and P. G. Guyenet, "Angiotensin II decreases a resting $K^+$ conductance in rat bulbospinal neurons of the c1 area," *Circulatiob Research*, vol. 78, pp. 274–282, 1996.

[9] B. Ogunnaike and W. H. Ray, *Process dynamics, Modeling and Control*. New York: Oxford University Press, 1995.
